# A Recurrent Neural Network for Generation of Ocular Saccades

**Lina L.E. Massone**
Department of Physiology
Department of Electrical Engineering and Computer Science
Northwestern University
303 E. Chicago Avenue, Chicago, Il 60611

## Abstract

This paper presents a neural network able to control saccadic movements. The input to the network is a specification of a stimulation site on the collicular motor map. The output is the time course of the eye position in the orbit (horizontal and vertical angles). The units in the network exhibit a one-to-one correspondance with neurons in the intermediate layer of the superior colliculus (collicular motor map), in the brainstem and with oculomotor neurons. Simulations carried out with this network demonstrate its ability to reproduce in a straightforward fashion many experimental observations.

## 1. INTRODUCTION

It is known that the superior colliculus (SC) plays an important role in the control of eye movements (Schiller et al. 1980). Electrophysiological studies (Cynader and Berman 1972, Robinson 1972) showed that the intermediate layer of SC is topographically organized into a motor map. The location of active neurons in this area was found to be related to the oculomotor error (i.e. how far the eyes are from the target) and their firing rate to saccade velocity (Roher et al. 1987, Berthoz et al. 1987). Neurons in the rostral area of the motor map, the so-called fixation neurons, tend to become active when the eyes are on target (Munoz and Wurtz 1992) and they can provide a gating mechanism to

arrest the movement (Guitton 1992). SC sends signals to the brainstem whose circuitry translates them into commands to the oculomotor neurons that innervate the eye muscles (Robinson 1981).

This paper presents a recurrent neural network that performs a spatio-temporal transformation from a stimulation site on the collicular motor map and an eye movement. The units in the network correspond to neurons in the intermediate layer of the colliculus, neurons in the brainstem and to oculomotor neurons.

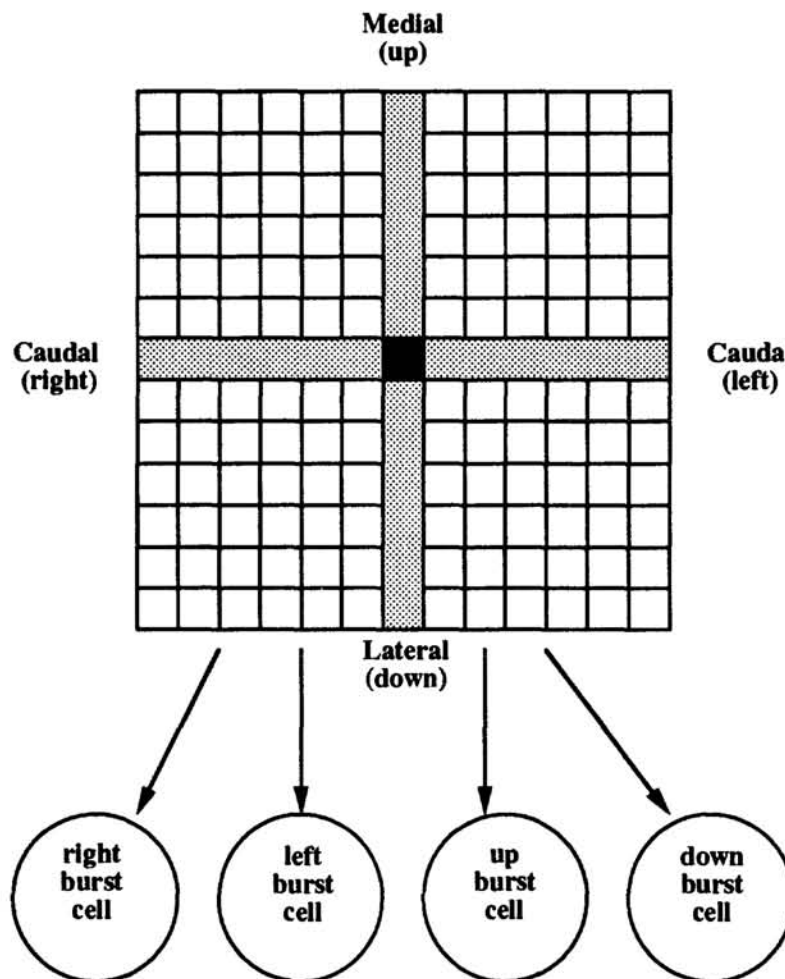

Figure 1: An array of units that represents the collicular motor map. The dark square represents the fixation area. The units in the array project to four units that represent burst cells devoted to process rightward, leftward, upward and downward saccades.

The network was built entirely on anatomical and physiological observations. Specifically, the following assumptions were used: (1) The activity on the collicular motor map shifts towards the fixation area during movement (Munoz et al. 1991, Droulez and Berthoz 1991). (2) The output of the superior colliculus is a vectorial velocity signal

that is the sum of the contributions from each active collicular neuron. (3) Such signal is decomposed into horizontal velocity and vertical velocity by a topographic and graded connectivity pattern from SC to the burst cells in the brainstem. (4) The computation performed from the burst-cells level down to the actual eye movement is carried out according to the push-pull arrangement proposed by Robinson (1981). (5) The activity on the collicular motor map is shifted by signals that represent the eye velocity. Efferent copies of the horizontal and vertical eye velocities are fed back onto the collicular map in order to implement the activity shift.

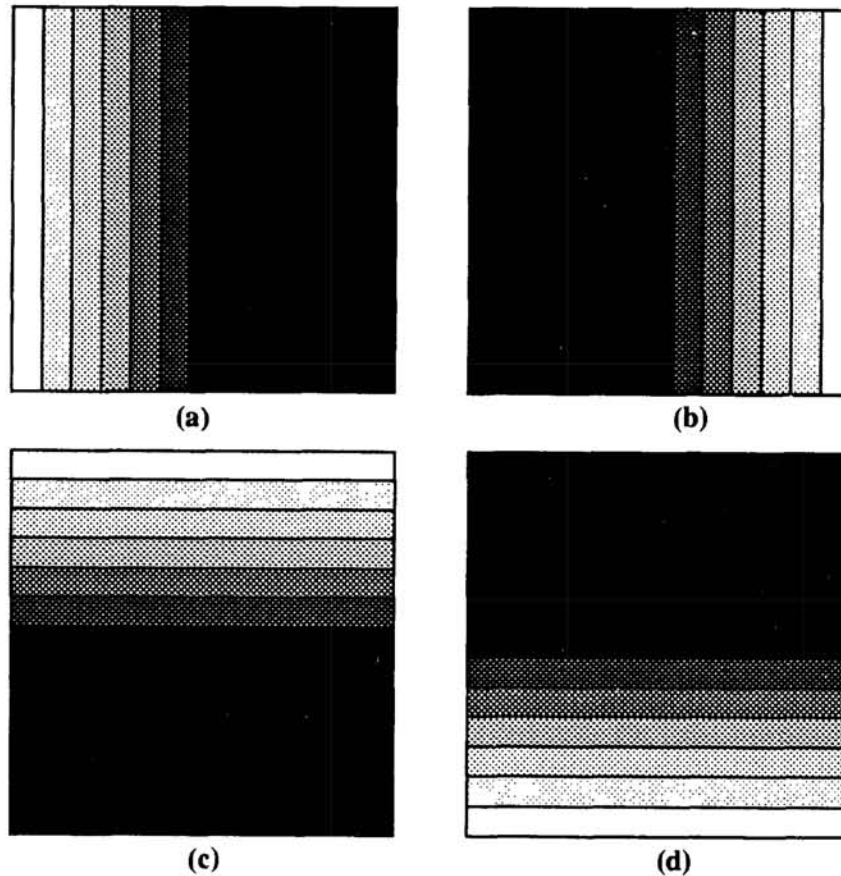

Figure 2: The topographic and graded pattern of connectivity from the collicular array to the four burst cells. Black means no connection, brighter colors represent larger weight values. (a) To the right cell. (b) To the left cell. (c) To the up cell. (d) To the down cell.

Simulations conducted with such a system (Massone submitted) demonstrated the network's ability to reproduce a number of experimental observations. Namely the network can: (1) Spontaneously produce oblique saccades whose curvature varies with the ratio between the horizontal and vertical components of the motor error.(2) Automatically hold the eye position in the orbit at the end of a saccade by exploiting the internal dynamic of the network. (3) Continuously produce efferent copies of the movements

without the need for reset signals. (4) Account for the outcome of the lidocaine experiment (Lee et al. 1988) without assuming a population averaging mechanism.

Section 2 describes the network architecture. A more detailed description of the network, it mechanisms and physiological ground as well as a number of simulation results can be found in Massone (submitted).

## 2. THE NETWORK

The network input layer is a bidimensional array of linear units that represent neurons in the collicular motor map. The array is topographically arranged as shown in Figure 1. Activity along the caudal axis produces horizontal saccades in a contralateral fashion, activity along the medio-lateral axis produces vertical saccades, activity in the rest of the array produces oblique saccades. The dark square in the center (rostral area) represents the fixation area. The units in this array project to four logistic units that represent two pairs of burst cells, one pair devoted to control horizontal movements, one pair devoted to control vertical movements. The pattern of connectivity between the collicular array and the units that represent the burst cells is qualitatively shown in Figure 2. The value of the weights of such connections increases exponentially when one moves from the center towards the periphery of the array. The fixation area projects to four other units that represent the so-called omnipause neurons. These units send a gating signal to the burst-cells units and are responsible for arresting the movement when the eyes are on target. i.e. when the activity in the input array reaches the center. Each pair of burst-cells units project to the network shown in Figure 3. This network is a computational version of the push-pull arrangement proposed by Robinson (1981). The bottom part of the network represents the oculomotor plant, the top part represents the brainstem circuitry and the oculomotor neurons. The weights in the bottom part of the network were derived by splitting into two equations the differential equation proposed by Robinson (1981) to describe the behavior of the oculomotor plant under a combined motorneuron input R.

$$R_1 = k\theta_1 + r\frac{d\theta_1}{dt}$$

$$R_2 = k\theta_2 + r\frac{d\theta_2}{dt}$$

R1 and R2 are the firing rates of the agonist and antagonist motorneurons, $\theta1$ and $\theta2$ are the components of the eye position due to motions in opposite directions (e.g. left and right), k is the eye stiffness and r is the eye viscosity.

The weights in the top part of the network were analytically computed from the weights in the bottom part of the network by imposing the following constraints: (1) The difference between $\theta1$ and $\theta2$ must produce the correct $\theta$. (2) The output of the neural integrators must be an efferent copy of the eye movement. (3) The output of the motorneurons must hold the eye at the current orbital position when the burst-cells units are shut off by the gating action of the omnipause cells. Efferent copies of the horizontal and vertical eye velocities were computed by differentiating the output of the neural

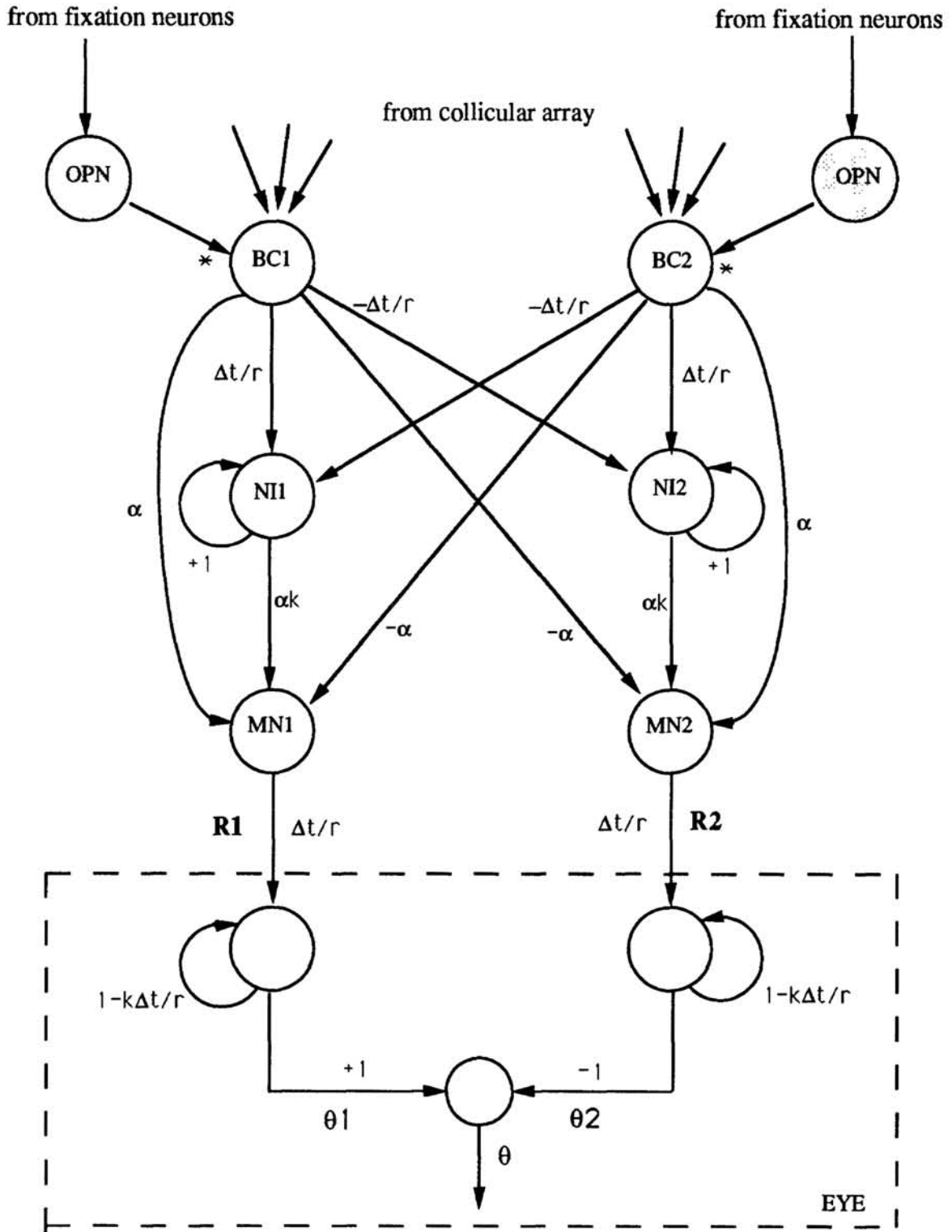

Figure 3: The recurrent network used to control eye movements in one direction, e.g. horizontal. An identical network is required to control vertical movements. OPN: omnipause neurons. BC1, BC2: burst cells. NI1, NI2: neural integrators. MN1, MN2: motor neurons. The architecture is based on Robinson's push-pull arrangement. k=4.0, r=0.95, α=0.5, Δt=1 msec.

integrators. These signals were recurrently fed back onto the input array and made the activity in the array shift towards the fixation area. This architecture assumes that the output of the collicular array represents saccade velocity. The network is started by selecting one unit in the input array, i.e. a "stimulation" site. When the unit is selected, a square area centered at that unit becomes active with a gaussian activity profile (Ottes et al. 1986, Munoz and Guitton 1991). At the time the input units are activated the eye starts moving and, as a consequence of the velocity feedback the activity on the input array starts shifting. The movement is arrested when the fixation area becomes activated. The activity of all units in the network represents neurons firing rates and is expressed in spikes/second.

Figure 4 shows the response of the network when the collicular array is stimulated at two sites sequentially. Each site causes an oblique saccade with unequal components. Stimulation number 1 brings the eye up and to the right, stimulation number 2 brings the eye back to the initial position. Fixation is maintained for a while inbetween stimulations and at the end of the two movements. The resulting trajectories in the movement plane (vertical angle versus horizontal angle) demonstrate the ability of the network to (i) maintain the eye position in the orbit when the burst cells activation is set to zero by the gating action of the omnipause neurons, (ii) produce curved trajectories with opposite curvatures when the eye moves back and forth between the same two angular positions. None of the units in the network is ever reset between saccades; because of the push-pull arrangement, when the activity of one neural integrator increases, the activity of the antagonist integrator decreases. This mechanism ensures that their activity does not grow indefinetely.

# 3. CONCLUSIONS

In this paper I presented an anatomically and physiologically inspired network able to control saccadic movements and to reproduce the outcome of some experimental observations. The results of simulations carried out with this network can be found in Massone (submitted). This work is currently being extended to (i) modeling the activity shift phenomenon as the relaxation of a dynamical system to its equilibrium configuration rather than as a feedback-driven mechanism, (ii) studying the role of the collicular output signals in the calibration and accuracy of arm movements (Massone 1992).

## Acknowledgements

This work was supported by the National Science Foundation, grant BCS-9113455 to the author.

## References

Berthoz A., Grantyn A., Droulez J. (1987) Some collicular neurons code saccadic eye velocity, *Neuroscience Letters*, 72, 289-294.

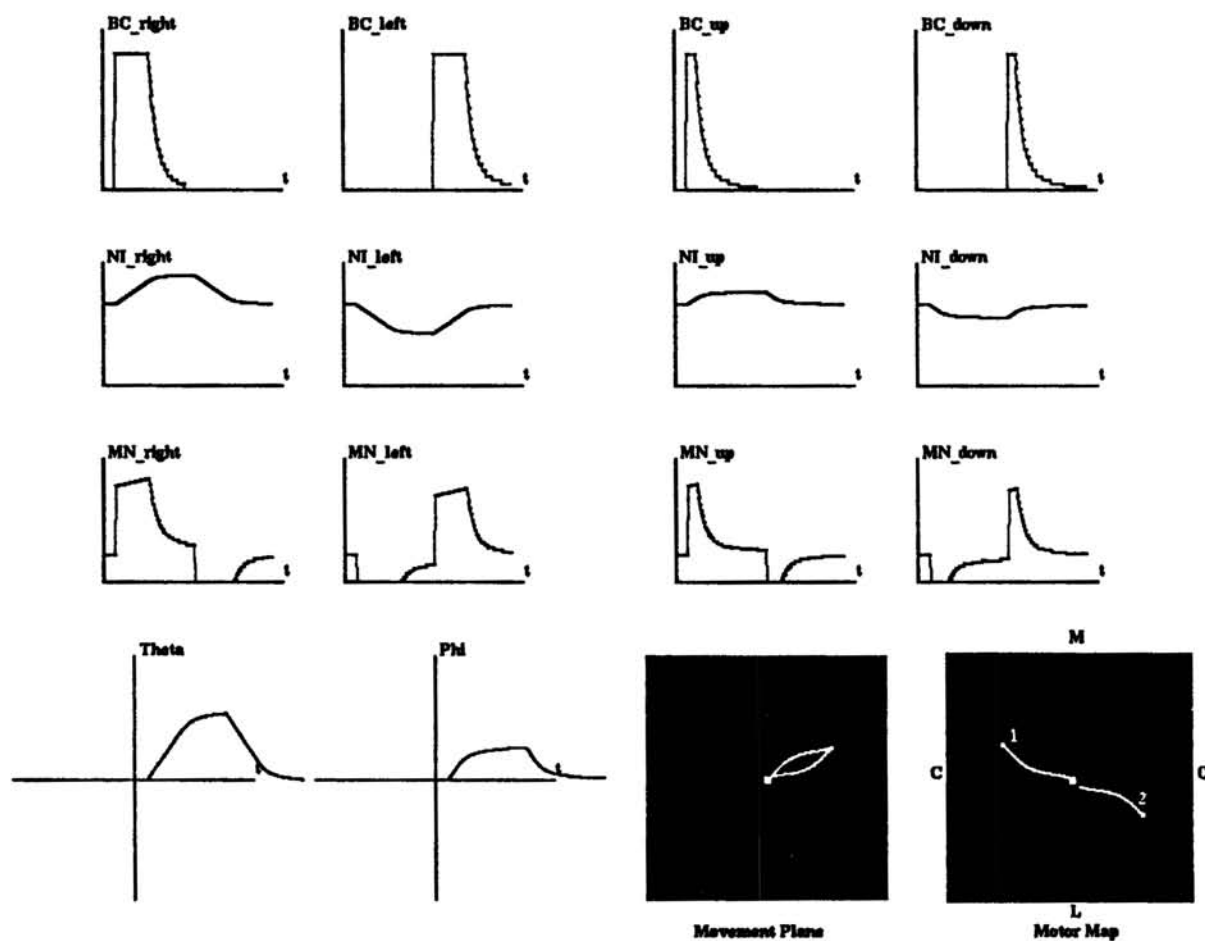

Figure 4: The response of the network to two sequential stimulations that produce two oblique saccades with unequal components.

Cynader M., Berman N. (1972) Receptive-field organization of monkey superior colliculus, *Journal of Neurophysiology*, 35, 187-201.

Droulez J., Berthoz A. (1991) The concept of dynamic memory in sensorimotor control, in *Motor Control Concepts and Issues*, Humphrey D.R. and Freund H.J. Eds., J. Whiley and Sons, 137-161.

Guitton D. (1992) Control of eye-head coordination during orienting gaze shifts, *Trends in Neuroscience*, 15(5), 174-179.

Lee C., Roher W.H., Sparks D.L. (1988) Population coding of saccadic eye movements by neurons in the superior colliculus, *Nature*, 332, 357-360.

Massone L. E. (1992) A biologically-inspired architecture for reactive motor control, in *Neural Networks for Control*, G. Beckey and K. Goldberg Eds., Kluwer Academic Publishers, 1992.

Massone L.E. (submitted) A velocity-based model for control of ocular saccades, *Neural Computation*.

Munoz D.P., Pellisson D., Guitton D. (1991) Movement of Neural Activity on the Superior Colliculus Motor Map during Gaze Shifts, *Science*, 251, 1358-1360.

Munoz D.P., Guitton D. (1991) Gaze control by the tecto-reticulo-spinal system in the head-free cat. II. Sustained discharges coding gaze position error, *Journal of Neurophysiology*, 66, 1624-1641.

Munoz D.P., Wurtz R.H. (1992) Role of the rostral superior colliculus in active visual fixation and execution of express saccades, *Journal of Neurophysiology*, 67, 1000-1002.

Ottes F.P., Van Gisbergen J.A.M., Eggermont J.J. (1986) Visuomotor fields of the superior colliculus: a quantitative model, *Vision Research*, 26, 857-873.

Robinson D.A. (1972) Eye movements evoked by collicular stimulation in the alert monkey, *Vision Research*, 12, 1795-1808.

Robinson D.A. (1981) Control of eye movements, in *Handbook of Physiology - The Nervous System II*, V.B. Brooks Ed., 1275-1320.

Roher W.H., White J.M., Sparks D.L. (1987) Saccade-related burst cells in the superior colliculus: relationship of activity with saccade velocity, *Society of Neuroscience Abstracts*, 13, 1092.